# ARA*: Anytime A* with Provable Bounds on Sub-Optimality

**Maxim Likhachev, Geoff Gordon and Sebastian Thrun**
School of Computer Science
Carnegie Mellon University
Pittsburgh, PA 15213
{maxim+, ggordon, thrun}@cs.cmu.edu

## Abstract

In real world planning problems, time for deliberation is often limited. Anytime planners are well suited for these problems: they find a feasible solution quickly and then continually work on improving it until time runs out. In this paper we propose an anytime heuristic search, ARA*, which tunes its performance bound based on available search time. It starts by finding a suboptimal solution quickly using a loose bound, then tightens the bound progressively as time allows. Given enough time it finds a provably optimal solution. While improving its bound, ARA* reuses previous search efforts and, as a result, is significantly more efficient than other anytime search methods. In addition to our theoretical analysis, we demonstrate the practical utility of ARA* with experiments on a simulated robot kinematic arm and a dynamic path planning problem for an outdoor rover.

## 1  Introduction

Optimal search is often infeasible for real world problems, as we are given a limited amount of time for deliberation and want to find the best solution given the time provided. In these conditions anytime algorithms [9, 2] prove to be useful as they usually find a first, possibly highly suboptimal, solution very fast and then continually work on improving the solution until allocated time expires. Unfortunately, they can rarely provide bounds on the sub-optimality of their solutions unless the cost of an optimal solution is already known. Even less often can these algorithms control their sub-optimality. Providing sub-optimality bounds is valuable, though: it allows one to judge the quality of the current plan, decide whether to continue or preempt search based on the current sub-optimality, and evaluate the quality of past planning episodes and allocate time for future planning episodes accordingly. Control over the sub-optimality bounds helps in adjusting the tradeoff between computation and plan quality.

A* search with inflated heuristics (actual heuristic values are multiplied by an inflation factor $\epsilon > 1$) is sub-optimal but proves to be fast for many domains [1, 5, 8] and also provides a bound on the sub-optimality, namely, the $\epsilon$ by which the heuristic is inflated [7]. To construct an anytime algorithm with sub-optimality bounds one could run a succession of these A* searches with decreasing inflation factors. This naive approach results in a series of solutions, each one with a sub-optimality factor equal to the corresponding inflation

factor. This approach has control over the sub-optimality bound, but wastes a lot of computation since each search iteration duplicates most of the efforts of the previous searches. One could try to employ incremental heuristic searches (e.g., [4]), but the sub-optimality bounds for each search iteration would no longer be guaranteed.

To this end we propose the ARA* (Anytime Repairing A*) algorithm, which is an *efficient* anytime heuristic search that also runs A* with inflated heuristics in succession but reuses search efforts from previous executions in such a way that the sub-optimality bounds are still satisfied. As a result, a substantial speedup is achieved by not re-computing the state values that have been correctly computed in the previous iterations. We show the efficiency of ARA* on two different domains. An evaluation of ARA* on a simulated robot kinematic arm with six degrees of freedom shows up to 6-fold speedup over the succession of A* searches. We also demonstrate ARA* on the problem of planning a path for a mobile robot that takes into account the robot's dynamics.

The only other anytime heuristic search known to us is Anytime A*, described in [8]. It also first executes an A* with inflated heuristics and then continues to improve a solution. However, the algorithm does not have control over its sub-optimality bound, except by selecting the inflation factor of the first search. Our experiments show that ARA* is able to decrease its bounds much more gradually and, moreover, does so significantly faster. Another advantage of ARA* is that it guarantees to examine each state at most once during its first search, unlike the algorithm of [8]. This property is important because it provides a bound on the amount of time before ARA* produces its first plan. Nevertheless, as mentioned later, [8] describes a number of very interesting ideas that are also applicable to ARA*.

## 2 The ARA* Algorithm

### 2.1 A* with Weighted Heuristic

Normally, A* takes as input a heuristic $h(s)$ which must be consistent. That is, $h(s) \leq c(s, s') + h(s')$ for any successor $s'$ of $s$ if $s \neq s_{goal}$ and $h(s) = 0$ if $s = s_{goal}$. Here $c(s, s')$ denotes the cost of an edge from $s$ to $s'$ and has to be positive. Consistency, in its turn, guarantees that the heuristic is admissible: $h(s)$ is never larger than the true cost of reaching the goal from $s$. Inflating the heuristic (that is, using $\epsilon * h(s)$ for $\epsilon > 1$) often results in much fewer state expansions and consequently faster searches. However, inflating the heuristic may also violate the admissibility property, and as a result, a solution is no longer guaranteed to be optimal. The pseudocode of A* with inflated heuristic is given in Figure 1 for easy comparison with our algorithm, ARA*, presented later.

A* maintains two functions from states to real numbers: $g(s)$ is the cost of the current path from the start node to $s$ (it is assumed to be $\infty$ if no path to $s$ has been found yet), and $f(s) = g(s) + \epsilon * h(s)$ is an estimate of the total distance from start to goal going through $s$. A* also maintains a priority queue, *OPEN*, of states which it plans to expand. The *OPEN* queue is sorted by $f(s)$, so that A* always expands next the state which appears to be on the shortest path from start to goal. A* initializes the *OPEN* list with the start state, $s_{start}$ (line 02). Each time it expands a state $s$ (lines 04-11), it removes $s$ from *OPEN*. It then updates the $g$-values of all of $s$'s neighbors; if it decreases $g(s')$, it inserts $s'$ into *OPEN*. A* terminates as soon as the goal state is expanded.

```
01 g(s_start) = 0; OPEN = ∅;
02 insert s_start into OPEN with f(s_start) = ε * h(s_start);
03 while(s_goal is not expanded)
04    remove s with the smallest f-value from OPEN;
05    for each successor s' of s
06       if s' was not visited before then
07          f(s') = g(s') = ∞;
08       if g(s') > g(s) + c(s, s')
09          g(s') = g(s) + c(s, s');
10          f(s') = g(s') + ε * h(s');
11          insert s' into OPEN with f(s');
```

**Figure 1**: A* with heuristic weighted by $\epsilon \geq 1$

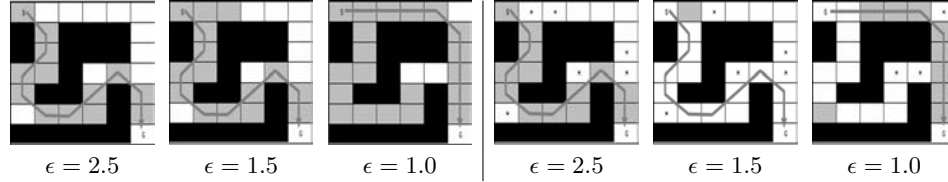

| $\epsilon = 2.5$ | $\epsilon = 1.5$ | $\epsilon = 1.0$ | $\epsilon = 2.5$ | $\epsilon = 1.5$ | $\epsilon = 1.0$ |

**Figure 2**: Left three columns: A* searches with decreasing $\epsilon$. Right three columns: the corresponding ARA* search iterations.

Setting $\epsilon$ to 1 results in standard A* with an uninflated heuristic; the resulting solution is guaranteed to be optimal. For $\epsilon > 1$ a solution can be sub-optimal, but the sub-optimality is bounded by a factor of $\epsilon$: the length of the found solution is no larger than $\epsilon$ times the length of the optimal solution [7].

The left three columns in Figure 2 show the operation of the A* algorithm with a heuristic inflated by $\epsilon = 2.5$, $\epsilon = 1.5$, and $\epsilon = 1$ (no inflation) on a simple grid world. In this example we use an eight-connected grid with black cells being obstacles. S denotes a start state, while G denotes a goal state. The cost of moving from one cell to its neighbor is one. The heuristic is the larger of the x and y distances from the cell to the goal. The cells which were expanded are shown in grey. (A* can stop search as soon as it is about to expand a goal state without actually expanding it. Thus, the goal state is not shown in grey.) The paths found by these searches are shown with grey arrows. The A* searches with inflated heuristics expand substantially fewer cells than A* with $\epsilon = 1$, but their solution is sub-optimal.

## 2.2   ARA*: Reuse of Search Results

ARA* works by executing A* multiple times, starting with a large $\epsilon$ and decreasing $\epsilon$ prior to each execution until $\epsilon = 1$. As a result, after each search a solution is guaranteed to be within a factor $\epsilon$ of optimal. Running A* search from scratch every time we decrease $\epsilon$, however, would be very expensive. We will now explain how ARA* reuses the results of the previous searches to save computation. We first explain the ImprovePath function (left column in Figure 3) that recomputes a path for a given $\epsilon$. In the next section we explain the Main function of ARA* (right column in Figure 3) that repetitively calls the ImprovePath function with a series of decreasing $\epsilon$s.

Let us first introduce a notion of *local inconsistency* (we borrow this term from [4]). A state is called locally inconsistent every time its $g$-value is decreased (line 09, Figure 1) and until the next time the state is expanded. That is, suppose that state $s$ is the best predecessor for some state $s'$: that is, $g(s') = \min_{s'' \in pred(s')}(g(s'') + c(s'', s')) = g(s) + c(s, s')$. Then, if $g(s)$ decreases we get $g(s') > \min_{s'' \in pred(s')}(g(s'') + c(s'', s'))$. In other words, the decrease in $g(s)$ introduces a local inconsistency between the $g$-value of $s$ and the $g$-values of its successors. Whenever $s$ is expanded, on the other hand, the inconsistency of $s$ is corrected by re-evaluating the $g$-values of the successors of $s$ (line 08-09, Figure 1). This in turn makes the successors of $s$ locally inconsistent. In this way the local inconsistency is propagated to the children of $s$ via a series of expansions. Eventually the children no longer rely on $s$, none of their $g$-values are lowered, and none of them are inserted into the *OPEN* list. Given this definition of local inconsistency it is clear that the *OPEN* list consists of exactly all locally inconsistent states: every time a $g$-value is lowered the state is inserted into *OPEN*, and every time a state is expanded it is removed from *OPEN* until the next time its $g$-value is lowered. Thus, the *OPEN* list can be viewed as a set of states from which we need to propagate local inconsistency.

A* with a consistent heuristic is guaranteed not to expand any state more than once. Setting $\epsilon > 1$, however, may violate consistency, and as a result A* search may re-expand states multiple times. It turns out that if we restrict each state to be expanded no more than once, then the sub-optimality bound of $\epsilon$ still holds. To implement this restriction we check any state whose $g$-value is lowered and insert it into *OPEN* only if it has not been previously expanded (line 10, Figure 3). The set of expanded states is maintained in the *CLOSED* variable.

**procedure fvalue($s$)**
01 return $g(s) + \epsilon * h(s)$;

**procedure ImprovePath()**
02 while(fvalue($s_{goal}$) $> \min_{s \in OPEN}$(fvalue($s$)))
03   remove $s$ with the smallest fvalue($s$) from $OPEN$;
04   $CLOSED = CLOSED \cup \{s\}$;
05   for each successor $s'$ of $s$
06     if $s'$ was not visited before then
07       $g(s') = \infty$;
08     if $g(s') > g(s) + c(s, s')$
09       $g(s') = g(s) + c(s, s')$;
10     if $s' \notin CLOSED$
11       insert $s'$ into $OPEN$ with fvalue($s'$);
12     else
13       insert $s'$ into $INCONS$;

**procedure Main()**
01' $g(s_{goal}) = \infty$; $g(s_{start}) = 0$;
02' $OPEN = CLOSED = INCONS = \emptyset$;
03' insert $s_{start}$ into $OPEN$ with fvalue($s_{start}$);
04' ImprovePath();
05' $\epsilon' = \min(\epsilon, g(s_{goal}) / \min_{s \in OPEN \cup INCONS}(\text{g}(s) + \text{h}(s)))$;
06' publish current $\epsilon'$-suboptimal solution;
07' while $\epsilon' > 1$
08'   decrease $\epsilon$;
09'   Move states from $INCONS$ into $OPEN$;
10'   Update the priorities for all $s \in OPEN$ according to fvalue($s$);
11'   $CLOSED = \emptyset$;
12'   ImprovePath();
13'   $\epsilon' = \min(\epsilon, g(s_{goal}) / \min_{s \in OPEN \cup INCONS}(\text{g}(s) + \text{h}(s)))$;
14'   publish current $\epsilon'$-suboptimal solution;

**Figure 3**: ARA*

With this restriction we will expand each state at most once, but *OPEN* may no longer contain all the locally inconsistent states. In fact, it will only contain the locally inconsistent states that have not yet been expanded. It is important, however, to keep track of *all* the locally inconsistent states as they will be the starting points for inconsistency propagation in the future search iterations. We do this by maintaining the set *INCONS* of all the locally inconsistent states that are not in *OPEN* (lines 12-13, Figure 3). Thus, the union of *INCONS* and *OPEN* is exactly the set of all locally inconsistent states, and can be used as a starting point for inconsistency propagation before each new search iteration.

The only other difference between the ImprovePath function and A* is the termination condition. Since the ImprovePath function reuses search efforts from the previous executions, $s_{goal}$ may never become locally inconsistent and thus may never be inserted into *OPEN*. As a result, the termination condition of A* becomes invalid. A* search, however, can also stop as soon as $f(s_{goal})$ is equal to the minimal $f$-value among all the states on *OPEN* list. This is the condition that we use in the ImprovePath function (line 02, Figure 3). It also allows us to avoid expanding $s_{goal}$ as well as possibly some other states with the same $f$-value. (Note that ARA* no longer maintains $f$-values as variables since in between the calls to the ImprovePath function $\epsilon$ is changed, and it would be prohibitively expensive to update the $f$-values of all the states. Instead, the fvalue($s$) function is called to compute and return the $f$-values only for the states in *OPEN* and $s_{goal}$.)

## 2.3 ARA*: Iterative Execution of Searches

We now introduce the main function of ARA* (right column in Figure 3) which performs a series of search iterations. It does initialization and then repetitively calls the ImprovePath function with a series of decreasing $\epsilon$s. Before each call to the ImprovePath function a new *OPEN* list is constructed by moving into it the contents of the set *INCONS*. Since *OPEN* list has to be sorted by the current $f$-values of states it is also re-ordered (lines 09'-10', Figure 3). Thus, after each call to the ImprovePath function we get a solution that is sub-optimal by at most a factor of $\epsilon$.

As suggested in [8] a sub-optimality bound can also be computed as the ratio between $g(s_{goal})$, which gives an upper bound on the cost of an optimal solution, and the minimum un-weighted $f$-value of a locally inconsistent state, which gives a lower bound on the cost of an optimal solution. (This is a valid sub-optimality bound as long as the ratio is larger than or equal to one. Otherwise, $g(s_{goal})$ is already equal to the cost of an optimal solution.) Thus, the actual sub-optimality bound for ARA* is computed as the minimum between $\epsilon$ and the ratio (lines 05' and 13', Figure 3). At first, one may also think of using this actual sub-optimality bound in deciding how to decrease $\epsilon$ between search iterations (e.g., setting $\epsilon$ to $\epsilon'$ minus a small delta). Experiments, however, seem to suggest that decreasing $\epsilon$ in small steps is still more beneficial. The reason is that a small decrease in $\epsilon$ often results in the improvement of the solution, despite the fact that the actual sub-optimality bound of the previous solution was already substantially less than the value of $\epsilon$. A large decrease in $\epsilon$, on the other hand, may often result in the expansion of too many states during the next search. (Another useful suggestion from [8], which we have not implemented in ARA*, is to prune *OPEN* so that it never contains a state whose un-weighted $f$-value is larger than

or equal to $g(s_{goal})$.)

Within each execution of the ImprovePath function we mainly save computation by not re-expanding the states which were locally consistent and whose $g$-values were already correct before the call to ImprovePath (Theorem 2 states this more precisely). For example, the right three columns in Figure 2 show a series of calls to the ImprovePath function. States that are locally inconsistent at the end of an iteration are shown with an asterisk. While the first call ($\epsilon = 2.5$) is identical to the A* call with the same $\epsilon$, the second call to the ImprovePath function ($\epsilon = 1.5$) expands only 1 cell. This is in contrast to 15 cells expanded by A* search with the same $\epsilon$. For both searches the sub-optimality factor, $\epsilon$, decreases from 2.5 to 1.5. Finally, the third call to the ImprovePath function with $\epsilon$ set to 1 expands only 9 cells. The solution is now optimal, and the total number of expansions is 23. Only 2 cells are expanded more than once across all three calls to the ImprovePath function. Even a single optimal search from scratch expands 20 cells.

### 2.4 Theoretical Properties of the Algorithm

We now present some of the theoretical properties of ARA*. For the proofs of these and other properties of the algorithm please refer to [6]. We use $g^*(s)$ to denote the cost of an optimal path from $s_{start}$ to $s$. Let us also define a *greedy path* from $s_{start}$ to $s$ as a path that is computed by tracing it backward as follows: start at $s$, and at any state $s_i$ pick a state $s_{i-1} = \arg\min_{s' \in pred(s_i)}(g(s') + c(s', s_i))$ until $s_{i-1} = s_{start}$.

**Theorem 1** *Whenever the ImprovePath function exits, for any state $s$ with $f(s) \leq \min_{s' \in \text{OPEN}}(f(s'))$, we have $g^*(s) \leq g(s) \leq \epsilon * g^*(s)$, and the cost of a greedy path from $s_{start}$ to $s$ is no larger than $g(s)$.*

The correctness of ARA* follows from this theorem: each execution of the ImprovePath function terminates when $f(s_{goal})$ is no larger than the minimum $f$-value in *OPEN*, which means that the greedy path from start to goal that we have found is within a factor $\epsilon$ of optimal. Since before each iteration $\epsilon$ is decreased, and it, in its turn, is an upper bound on $\epsilon'$, ARA* gradually decreases the sub-optimality bound and finds new solutions to satisfy the bound.

**Theorem 2** *Within each call to ImprovePath() a state is expanded at most once and only if it was locally inconsistent before the call to ImprovePath() or its g-value was lowered during the current execution of ImprovePath().*

The second theorem formalizes where the computational savings for ARA* search come from. Unlike A* search with an inflated heuristic, each search iteration in ARA* is guaranteed not to expand states more than once. Moreover, it also does not expand states whose $g$-values before a call to the ImprovePath function have already been correctly computed by some previous search iteration, unless they are in the set of locally inconsistent states already and thus need to update their neighbors (propagate local inconsistency).

## 3 Experimental Study

### 3.1 Robotic Arm

We first evaluate the performance of ARA* on simulated 6 and 20 degree of freedom (DOF) robotic arms (Figure 4). The base of the arm is fixed, and the task is to move its end-effector to the goal while navigating around obstacles (indicated by grey rectangles). An action is defined as a change of a global angle of any particular joint (i.e., the next joint further along the arm rotates in the opposite direction to maintain the global angle of the remaining joints.) We discretitize the workspace into 50 by 50 cells and compute a distance from each cell to the cell containing the goal while taking into account that some cells are occupied by obstacles. This distance is our heuristic. In order for the heuristic not to overestimate true costs, joint angles are discretitized so as to never move the end-effector by more than one cell in a single action. The resulting state-space is over 3 billion states for a 6 DOF robot arm and over $10^{26}$ states for a 20 DOF robot arm, and memory for states is allocated on demand.

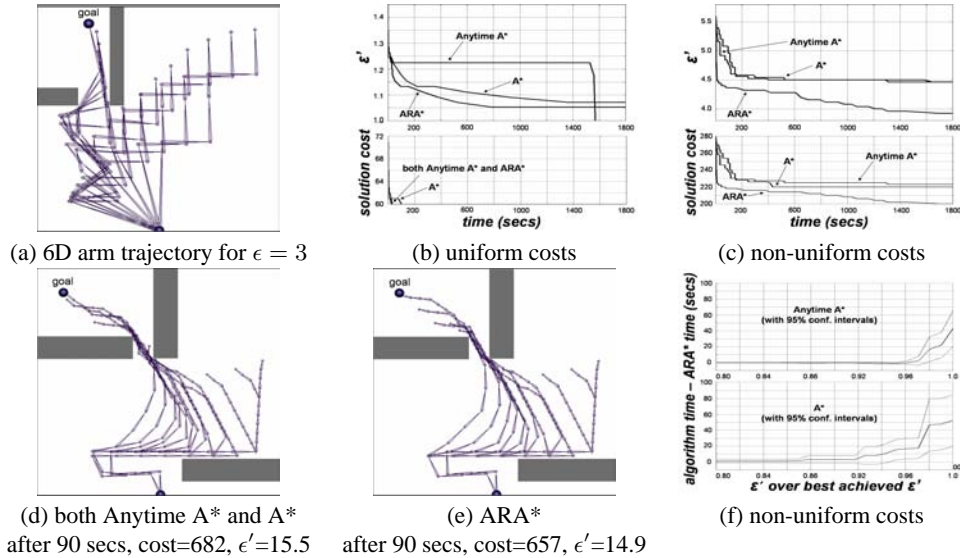

(a) 6D arm trajectory for $\epsilon = 3$  (b) uniform costs  (c) non-uniform costs

(d) both Anytime A* and A*  (e) ARA*  (f) non-uniform costs
after 90 secs, cost=682, $\epsilon'$=15.5  after 90 secs, cost=657, $\epsilon'$=14.9

**Figure 4**: Top row: 6D robot arm experiments. Bottom row: 20D robot arm experiments (the trajectories shown are downsampled by 6). Anytime A* is the algorithm in [8].

Figure 4a shows the planned trajectory of the robot arm after the initial search of ARA* with $\epsilon = 3.0$. This search takes about 0.05 secs. (By comparison, a search for an optimal trajectory is infeasible as it runs out of memory very quickly.) The plot in Figure 4b shows that ARA* improves both the quality of the solution and the bound on its sub-optimality faster and in a more gradual manner than either a succession of A* searches or Anytime A* [8]. In this experiment $\epsilon$ is initially set to 3.0 for all three algorithms. For all the experiments in this section $\epsilon$ is decreased in steps of 0.02 (2% sub-optimality) for ARA* and a succession of A* searches. Anytime A* does not control $\epsilon$, and in this experiment it apparently performs a lot of computations that result in a large decrease of $\epsilon$ at the end. On the other hand, it does reach the optimal solution first this way. To evaluate the expense of the anytime property of ARA* we also ran ARA* and an optimal A* search in a slightly simpler environment (for the optimal search to be feasible). Optimal A* search required about 5.3 mins (2,202,666 state expanded) to find an optimal solution, while ARA* required about 5.5 mins (2,207,178 state expanded) to decrease $\epsilon$ in steps of 0.02 from 3.0 until a provably optimal solution was found (about 4% overhead).

While in the experiment for Figure 4b all the actions have the same cost, in the experiment for Figure 4c actions have non-uniform costs: changing a joint angle closer to the base is more expensive than changing a higher joint angle. As a result of the non-uniform costs our heuristic becomes less informative, and so search is much more expensive. In this experiment we start with $\epsilon = 10$, and run all algorithms for 30 minutes. At the end, ARA* achieves a solution with a substantially smaller cost (200 vs. 220 for the succession of A* searches and 223 for Anytime A*) and a better sub-optimality bound (3.92 vs. 4.46 for both the succession of A* searches and Anytime A*). Also, since ARA* controls $\epsilon$ it decreases the cost of the solution gradually. Reading the graph differently, ARA* reaches a sub-optimality bound $\epsilon' = 4.5$ after about 59 thousand expansions and 11.7 secs, while the succession of A* searches reaches the same bound after 12.5 million expansions and 27.4 minutes (about 140-fold speedup by ARA*) and Anytime A* reaches it after over 4 million expansions and 8.8 minutes (over 44-fold speedup by ARA*). Similar results hold when comparing the amount of work each of the algorithms spend on obtaining a solution of cost 225. While Figure 4 shows execution time, the comparison of states expanded (not shown) is almost identical. Additionally, to demonstrate the advantage of ARA* expanding each state no more than once per search iteration, we compare the first searches of ARA* and Anytime A*: the first search of ARA* performed 6,378 expansions, while Anytime A* performed 8,994 expansions, mainly because some of the states were expanded up to

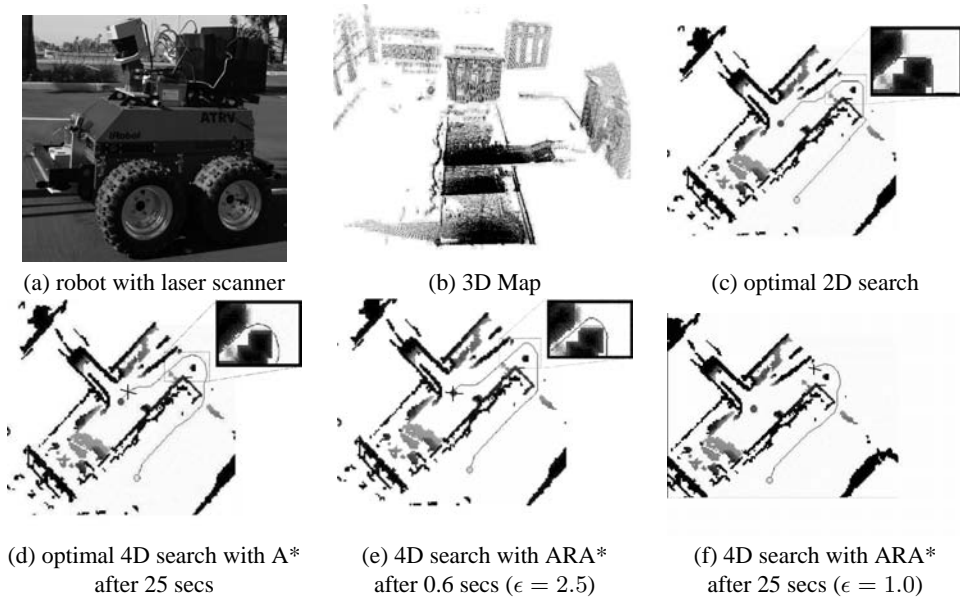

| (a) robot with laser scanner | (b) 3D Map | (c) optimal 2D search |
|---|---|---|
| (d) optimal 4D search with A* after 25 secs | (e) 4D search with ARA* after 0.6 secs ($\epsilon = 2.5$) | (f) 4D search with ARA* after 25 secs ($\epsilon = 1.0$) |

**Figure 5**: outdoor robot navigation experiment (cross shows the position of the robot)

seven times before a first solution was found.

Figures 4d-f show the results of experiments done on a 20 DOF robot arm, with actions that have non-uniform costs. All three algorithms start with $\epsilon = 30$. Figures 4d and 4e show that in 90 seconds of planning the cost of the trajectory found by ARA* and the sub-optimality bound it can guarantee is substantially smaller than for the other algorithms. For example, the trajectory in Figure 4d contains more steps and also makes one extra change in the angle of the third joint from the base of the arm (despite the fact that changing lower joint angles is very expensive) in comparison to the trajectory in Figure 4e. The graph in Figure 4f compares the performance of the three algorithms on twenty randomized environments similar to the environment in Figure 4d. The environments had random goal locations, and the obstacles were slid to random locations along the outside walls. The graph shows the additional time the other algorithms require to achieve the same sub-optimality bound that ARA* does. To make the results from different environments comparable we normalize the bound by dividing it by the maximum of the best bounds that the algorithms achieve before they run out of memory. Averaging over all environments, the time for ARA* to achieve the best bound was 10.1 secs. Thus, the difference of 40 seconds at the end of the Anytime A* graph corresponds to an overhead of about a factor of 4.

## 3.2  Outdoor Robot Navigation

For us the motivation for this work was efficient path-planning for mobile robots in large outdoor environments, where optimal trajectories involve fast motion and sweeping turns at speed. In such environments it is particularly important to take advantage of the robot's momentum and find dynamic rather than static plans. We use a 4D state space: $xy$ position, orientation, and velocity. High dimensionality and large environments result in very large state-spaces for the planner and make it computationally infeasible for the robot to plan optimally every time it discovers new obstacles or modelling errors. To solve this problem we built a two-level planner: a 4D planner that uses ARA*, and a fast 2D $(x, y)$ planner that uses A* search and whose results serve as the heuristic for the 4D planner.[1]

In Figure 5 we show the robot we used for navigation and a 3D laser scan [3] constructed by the robot of the environment we tested our system in. The scan is converted into a map of the environment (Figure 5c, obstacles shown in black). The size of the environment is 91.2 by 94.4 meters, and the map is discretitized into cells of 0.4 by 0.4 meters. Thus, the 2D state-space consists of 53808 states. The 4D state space has over 20 million states. The robot's initial state is the upper circle, while its goal is the lower circle. To ensure safe operation we created a buffer zone with high costs around each obstacle. The squares in the upper-right corners of the figures show a magnified fragment of the map with grayscale proportional to cost. The 2D plan (Figure 5c) makes sharp 45 degree turns when going around the obstacles, requiring the robot to come to complete stops. The optimal 4D plan results in a wider turn, and the velocity of the robot remains high throughout the whole trajectory. In the first plan computed by ARA* starting at $\epsilon = 2.5$ (Figure 5e) the trajectory is much better than the 2D plan, but somewhat worse than the optimal 4D plan.

The time required for the optimal 4D planner was 11.196 secs, whereas the time for the 4D ARA* planner to generate the plan in Figure 5e was 556ms. As a result, the robot that runs ARA* can start executing its plan much earlier. A robot running the optimal 4D planner would still be near the beginning of its path 25 seconds after receiving a goal location (Figure 5d). In contrast, in the same amount of time the robot running ARA* has advanced much further (Figure 5f), and its plan by now has converged to optimal ($\epsilon$ has decreased to 1).

## 4   Conclusions

We have presented the first anytime heuristic search that works by continually decreasing a sub-optimality bound on its solution and finding new solutions that satisfy the bound on the way. It executes a series of searches with decreasing sub-optimality bounds, and each search tries to reuse as much as possible of the results from previous searches. The experiments show that our algorithm is much more efficient than any of the previous anytime searches, and can successfully solve large robotic planning problems.

### Acknowledgments

This work was supported by AFRL contract F30602–01–C–0219, DARPA's MICA program.

## Footnotes

[1]To interleave search with the execution of the best plan so far we perform 4D search backward. That is, the start of the search, $s_{start}$, is the actual goal state of the robot, while the goal of the search, $s_{goal}$, is the current state of the robot. Thus, $s_{start}$ does not change as the robot moves and the search tree remains valid in between search iterations. Since heuristics estimate the distances to $s_{goal}$ (the robot position) we have to recompute them during the reorder operation (line 10', Figure 3).

## References

[1] B. Bonet and H. Geffner. Planning as heuristic search. *Artificial Intelligence*, 129(1-2):5–33, 2001.

[2] T. L. Dean and M. Boddy. An analysis of time-dependent planning. In *Proc. of the National Conference on Artificial Intelligence (AAAI)*, 1988.

[3] D. Haehnel. Personal communication, 2003.

[4] S. Koenig and M. Likhachev. Incremental A*. In *Advances in Neural Information Processing Systems (NIPS) 14*. Cambridge, MA: MIT Press, 2002.

[5] R. E. Korf. Linear-space best-first search. *Artificial Intelligence*, 62:41–78, 1993.

[6] M. Likhachev, G. Gordon, and S. Thrun. ARA*: Formal Analysis. Tech. Rep. CMU-CS-03-148, Carnegie Mellon University, Pittsburgh, PA, 2003.

[7] J. Pearl. *Heuristics: Intelligent Search Strategies for Computer Problem Solving*. Addison-Wesley, 1984.

[8] R. Zhou and E. A. Hansen. Multiple sequence alignment using A*. In *Proc. of the National Conference on Artificial Intelligence (AAAI)*, 2002. Student abstract.

[9] S. Zilberstein and S. Russell. Approximate reasoning using anytime algorithms. In *Imprecise and Approximate Computation*. Kluwer Academic Publishers, 1995.
